# Nonparametric Bayesian Texture Learning and Synthesis

**Long (Leo) Zhu**[1] **Yuanhao Chen**[2] **William Freeman**[1] **Antonio Torralba**[1]

[1]CSAIL, MIT        [2]Department of Statistics, UCLA

{leozhu, billf, antonio}@csail.mit.edu       yhchen@stat.ucla.edu

## Abstract

We present a nonparametric Bayesian method for texture learning and synthesis. A texture image is represented by a 2D Hidden Markov Model (2DHMM) where the hidden states correspond to the cluster labeling of textons and the transition matrix encodes their spatial layout (the compatibility between adjacent textons). The 2DHMM is coupled with the Hierarchical Dirichlet process (HDP) which allows the number of textons and the complexity of transition matrix grow as the input texture becomes irregular. The HDP makes use of Dirichlet process prior which favors regular textures by penalizing the model complexity. This framework (HDP-2DHMM) learns the texton vocabulary and their spatial layout jointly and automatically. The HDP-2DHMM results in a compact representation of textures which allows fast texture synthesis with comparable rendering quality over the state-of-the-art patch-based rendering methods. We also show that the HDP-2DHMM can be applied to perform image segmentation and synthesis. The preliminary results suggest that HDP-2DHMM is generally useful for further applications in low-level vision problems.

## 1 Introduction

Texture learning and synthesis are important tasks in computer vision and graphics. Recent attempts can be categorized into two different styles. The first style emphasizes the modeling and understanding problems and develops statistical models [1, 2] which are capable of representing texture using textons and their spatial layout. But the learning is rather sensitive to the parameter settings and the rendering quality and speed is still not satisfactory. The second style relies on patch-based rendering techniques [3, 4] which focus on rendering quality and speed, but forego the semantic understanding and modeling of texture.

This paper aims at texture understanding and modeling with fast synthesis and high rendering quality. Our strategy is to augment the patch-based rendering method [3] with nonparametric Bayesian modeling and statistical learning. We represent a texture image by a 2D Hidden Markov Model (2D-HMM) (see figure (1)) where the hidden states correspond to the cluster labeling of textons and the transition matrix encodes the texton spatial layout (the compatibility between adjacent textons). The 2D-HMM is coupled with the Hierarchical Dirichlet process (HDP) [5, 6] which allows the number of textons (i.e. hidden states) and the complexity of the transition matrix to grow as more training data is available or the randomness of the input texture becomes large. The Dirichlet process prior penalizes the model complexity to favor reusing clusters and transitions and thus regular texture which can be represented by compact models. This framework (HDP-2DHMM) discovers the semantic meaning of texture in an explicit way that the texton vocabulary and their spatial layout are learnt jointly and automatically (the number of textons is fully determined by HDP-2DHMM).

Once the texton vocabulary and the transition matrix are learnt, the synthesis process samples the latent texton labeling map according to the probability encoded in the transition matrix. The final

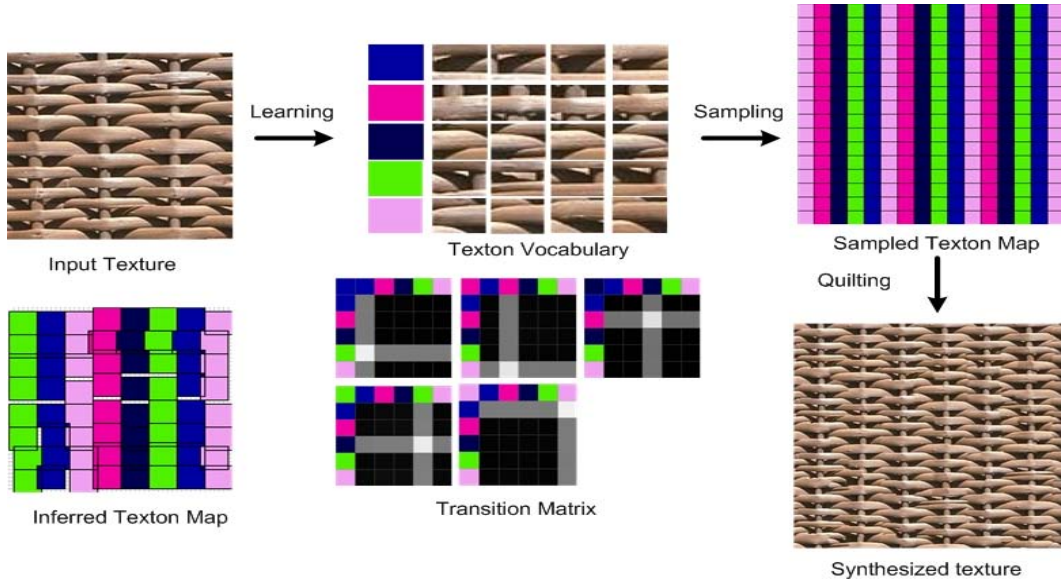

Figure 1: The flow chart of texture learning and synthesis. The colored rectangles correspond to the index (labeling) of textons which are represented by image patches. The texton vocabulary shows the correspondence between the color (states) and the examples of image patches. The transition matrices show the probability (indicated by the intensity) of generating a new state (coded by the color of the top left corner rectangle), given the states of the left and upper neighbor nodes (coded by the top and left-most rectangles). The inferred texton map shows the state assignments of the input texture. The top-right panel shows the sampled texton map according to the transition matrices. The last panel shows the synthesized texture using image quilting according to the correspondence between the sampled texton map and the texton vocabulary.

image is then generated by selecting the image patches based on the sampled texton labeling map. Here, image quilting [3] is applied to search and stitch together all the patches so that the boundary inconsistency is minimized. By contrast to [3], our method is only required to search a much smaller set of candidate patches within a local texton cluster. Therefore, the synthesis cost is dramatically reduced. We show that the HDP-2DHMM is able to synthesize texture in one second (25 times faster than image quilting) with comparable quality. In addition, the HDP-2DHMM is less sensitive to the patch size which has to be tuned over different input images in [3].

We also show that the HDP-2DHMM can be applied to perform image segmentation and synthesis. The preliminary results suggest that the HDP-2DHMM is generally useful for further applications in low-level vision problems.

## 2   Previous Work

Our primary interest is texture understanding and modeling. The FRAME model [7] provides a principled way to learn Markov random field models according to the marginal image statistics. This model is very successful in capturing stochastic textures, but may fail for more structured textures due to lack of spatial modeling. Zhu et al. [1, 2] extend it to explicitly learn the textons and their spatial relations which are represented by extra hidden layers. This new model is parametric (the number of texton clusters has to be tuned by hand for different texture images) and model selection which might be unstable in practice, is needed to avoid overfitting. Therefore, the learning is sensitive to the parameter settings. Inspired by recent progress in machine learning, we extend the nonparametric Bayesian framework of coupling 1D HMM and HDP [6] to deal with 2D texture image. A new model (HDP-2DHMM) is developed to learn texton vocabulary and spatial layouts jointly and automatically.

Since the HDP-2DHMM is designed to generate appropriate image statistics, but not pixel intensity, a patch-based texture synthesis technique, called image quilting [3], is integrated into our system to sample image patches. The texture synthesis algorithm has also been applied to image inpainting [8].

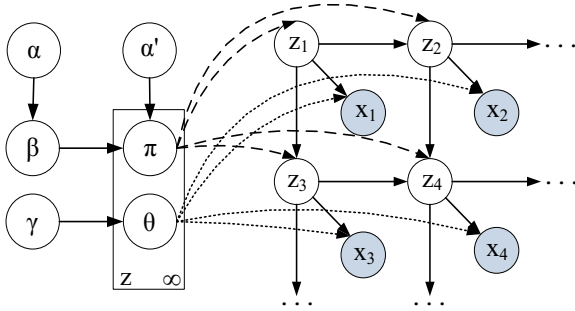

Figure 2: Graphical representation of the HDP-2DHMM. $\alpha, \alpha', \gamma$ are hyperparameters set by hand. $\beta$ are state parameters. $\theta$ and $\pi$ are emission and transition parameters, respectively. $i$ is the index of nodes in HMM. $L(i)$ and $T(i)$ are two nodes on the left and top of node $i$. $z_i$ are hidden states of node $i$. $x_i$ are observations (features) of the image patch at position $i$.

Malik et al. [9, 10] and Varma and Zisserman [11] study the filter representations of textons which are related to our implementations of visual features. But the interactions between textons are not explicitly considered. Liu et al. [12, 13] address texture understanding by discovering regularity without explicit statistical texture modeling.

Our work has partial similarities with the epitome [14] and jigsaw [15] models for non-texture images which also tend to model appearance and spatial layouts jointly. The major difference is that their models, which are parametric, cannot grow automatically as more data is available. Our method is closely related to [16] which is not designed for texture learning. They use hierarchical Dirichlet process, but the models and the image feature representations, including both the image filters and the data likelihood model, are different. The structure of 2DHMM is also discussed in [17]. Other work using Dirichlet prior includes [18, 19].

Tree-structured vector quantization [20] has been used to speed up existing image-based rendering algorithms. While this is orthogonal to our work, it may help us optimize the rendering speed. The meaning of "nonparametric" in this paper is under the context of Bayesian framework which differs from the non-Bayesian terminology used in [4].

## 3 Texture Modeling

### 3.1 Image Patches and Features

A texture image $I$ is represented by a grid of image patches $\{x_i\}$ with size of $24 \times 24$ in this paper where $i$ denotes the location. $\{x_i\}$ will be grouped into different textons by the HDP-2DHMM. We begin with a simplified model where the positions of textons represented by image patches are pre-determined by the image grid, and not allowed to shift. We will remove this constraint later.

Each patch $x_i$ is characterized by a set of filter responses $\{w_i^{l,h,b}\}$ which correspond to values $b$ of image filter response $h$ at location $l$. More precisely, each patch is divided into 6 by 6 cells (i.e. $l = 1..36$) each of which contains 4 by 4 pixels. For each pixel in cell $l$, we calculate 37 ($h = 1..37$) image filter responses which include the 17 filters used in [21], Difference of Gaussian (DOG, 4 filters), Difference of Offset Gaussian (DOOG, 12 filters ) and colors (R,G,B and L). $w_i^{l,h,b}$ equals one if the averaged value of filter responses of the 4*4 pixels covered by cell $l$ falls into bin $b$ (the response values are divided into 6 bins), and zero otherwise. Therefore, each patch $x_i$ is represented by 7992 ($= 37 * 36 * 6$) dimensional feature responses $\{w_i^{l,h,b}\}$ in total. We let $q = 1..7992$ denote the index of the responses of visual features.

It is worth emphasizing that our feature representation differs from standard methods [10, 2] where k-means clustering is applied to form visual vocabulary first. By contrast, we skip the clustering step and leave the learning of texton vocabulary together with spatial layout learning into the HDP-2DHMM which takes over the role of k-means.

### 3.2 HDP-2DHMM: Coupling Hidden Markov Model with Hierarchical Dirichlet Process

A texture is modeled by a 2D Hidden Markov Model (2DHMM) where the nodes correspond to the image patches $x_i$ and the compatibility is encoded by the edges connecting 4 neighboring nodes. See the graphical representation of 2DHMM in figure 2. For any node $i$, let $L(i), T(i), R(i), D(i)$ denote the four neighbors, left, upper, right and lower, respectively. We use $z_i$ to index the states

- $\beta \sim GEM(\alpha)$
- For each state $z \in \{1, 2, 3, ...\}$
  - $\theta_z \sim Dirichlet(\gamma)$
  - $\pi_{z_L} \sim DP(\alpha', \beta)$
  - $\pi_{z_T} \sim DP(\alpha', \beta)$
- For each pair of states $(z_L, z_T)$
  - $\pi_{z_L, z_T} \sim DP(\alpha', \beta)$
- For each node $i$ in the HMM
  - if $L(i) \neq \emptyset \ and \ T(i) \neq \emptyset$: $z_i|(z_{L(i)}, z_{T(i)}) \sim Multinomial(\pi_{z_L, z_T})$
  - if $L(i) \neq \emptyset \ and \ T(i) = \emptyset$: $z_i|z_{L(i)} \sim Multinomial(\pi_{z_L})$
  - if $L(i) = \emptyset \ and \ T(i) \neq \emptyset$: $z_i|z_{T(i)} \sim Multinomial(\pi_{z_T})$
  - $x_i \sim Multinomial(\theta_{z_i})$

Figure 3: HDP-2DHMM for texture modeling

of node $i$ which correspond to the cluster labeling of textons. The likelihood model $p(x_i|z_i)$ which specifies the probability of visual fetures is defined by multinomial distribution parameterized by $\theta_{z_i}$ specific to its corresponding hidden state $z_i$:

$$x_i \sim Multinomial(\theta_{z_i}) \tag{1}$$

where $\theta_{z_i}$ specify the weights of visual features.

For node $i$ which is connected to the nodes above and on the left (i.e. $L(i) \neq \emptyset \ and \ T(i) \neq \emptyset$), the probability $p(z_i|z_{L(i)}, z_{T(i)})$ of its state $z_i$ is only determined by the states $(z_{L(i)}, z_{T(i)})$ of the connected nodes. The distribution has a form of multinomial distribution parameterized by $\pi_{z_{L(i)}, z_{T(i)}}$:

$$z_i \sim Multinomial(\pi_{z_{L(i)}, z_{T(i)}}) \tag{2}$$

where $\pi_{z_{L(i)}, z_{T(i)}}$ encodes the transition matrix and thus the spatial layout of textons.

For the nodes which are on the top row or the left-most column (i.e. $L(i) = \emptyset \ or \ T(i) = \emptyset$), the distribution of their states are modeled by $Multinomial(\pi_{z_{L(i)}})$ or $Multinomial(\pi_{z_{T(i)}})$ which can be considered as simpler cases. We assume the top left corner can be sampled from any states according to the marginal statistics of states. Without loss of generality, we will skip the details of the boundary cases, but only focus on the nodes whose states should be determined by their top and left nodes jointly.

To make a nonparametric Bayesian representation, we need to allow the number of states $z_i$ countably infinite and put prior distributions over the parameters $\theta_{z_i}$ and $\pi_{z_{L(i)}, z_{T(i)}}$. We can achieve this by tying the 2DHMM together with the hierarchical Dirichlet process [5]. We define the prior of $\theta_z$ as a conjugate Dirichlet prior:

$$\theta_z \sim Dirichlet(\gamma) \tag{3}$$

where $\gamma$ is the concentration hyperparameter which controls how uniform the distribution of $\theta_z$ is (note $\theta_z$ specify weights of visual features): as $\gamma$ increases, it becomes more likely that the visual features have equal probability. Since the likelihood model $p(x_i|z_i)$ is of multinomial form, the posterior distribution of $\theta_z$ has a analytic form, still a Dirichlet distribtion.

The transition parameters $\pi_{z_L, z_T}$ are modeled by a hierarchical Dirichlet process (HDP):

$$\beta \sim GEM(\alpha) \tag{4}$$

$$\pi_{z_L, z_T} \sim DP(\alpha', \beta) \tag{5}$$

where we first draw global weights $\beta$ according to the stick-breaking prior distribution $GEM(\alpha)$. The stick-breaking weights $\beta$ specify the probability of state which are globally shared among all nodes. The stick-breaking prior produces exponentially decayed weights in expectation such that simple models with less representative clusters (textons) are favored, given few observations, but, there is always a low-probability that small clusters are created to capture details revealed by large, complex textures. The concentration hyperparameter $\alpha$ controls the sparseness of states: a larger $\alpha$ leads to more states. The prior of the transition parameter $\pi_{z_L, z_T}$ is modeled by a Dirichlet

process $DP(\alpha', \beta)$ which is a distribution over the other distribution $\beta$. $\alpha'$ is a hyperparameter which controls the variability of $\pi_{z_L, z_T}$ over different states across all nodes: as $\alpha'$ increases, the state transitions become more regular. Therefore, the HDP makes use of a Dirichlet process prior to place a soft bias towards simpler models (in terms of the number of states and the regularity of state transitions) which explain the texture.

The generative process of the HDP-2DHMM is described in figure (3). We now have the full representation of the HDP-2DHMM. But this simplified model does not allow the textons (image patches) to be shifted. We remove this constraint by introducing two hidden variables $(u_i, v_i)$ which indicate the displacements of textons associated with node $i$. We only need to adjust the correspondence between image features $x_i$ and hidden states $z_i$. $x_i$ is modified to be $x_{u_i, v_i}$ which refers to image features located at the position with displacement of $(u_i, v_i)$ to the position $i$. Random variables $(u_i, v_i)$ are only connected to the observation $x_i$ (not shown in figure 2). $(u_i, v_i)$ have a uniform prior, but are limited to the small neighborhood of $i$ (maximum 10% shift on one side).

## 4 Learning HDP-2DHMM

In a Bayesian framework, the task of learning HDP-2DHMM (also called Bayesian inference) is to compute the posterior distribution $p(\theta, \pi, z|x)$. It is trivial to sample the hidden variables $(u, v)$ because of their uniform prior. For simplicity, we skip the details of sampling $u, v$. Here, we present an inference procedure for the HDP-2DHMM that is based on Gibbs sampling. Our procedure alternates between two sampling stages: (i) sampling the state assignments $z$, (ii) sampling the global weights $\beta$. Given fixed values for $z, \beta$, the posterior of $\theta$ can be easily obtained by aggregating statistics of the observations assigned to each state. The posterior of $\pi$ is Dirichlet. For more details on Dirichlet processes, see [5].

We first instantiate a random hidden state labeling and then iteratively repeat the following two steps.

**Sampling $z$.** In this stage we sample a state for each node. The probability of node $i$ being assigned state $t$ is given by:

$$P(z_i = t|z^{-i}, \beta) \propto f_t^{-x_i}(x_i) P(z_i = t|z_{L(i)}, z_{T(i)})$$
$$\cdot P(z_{R(i)}|z_i = t, z_{T(R(i))}) P(z_{D(i)}|z_{L(D(i))}, z_i = t) \tag{6}$$

The first term $f_t^{-x_i}(x_i)$ denotes the posterior probability of observation $x_i$ given all other observations assigned to state $t$, and $z^{-i}$ denotes all state assignments except $z_i$. Let $n_{qt}$ be the number of observations of feature $w^q$ with state $t$. $f_t^{-x_i}(x_i)$ is calculated by:

$$f_t^{-x_i}(x_i) = \prod_q \left( \frac{n_{qt} + \gamma_q}{\sum_{q'} n_{q't} + \sum_{q'} \gamma_{q'}} \right)^{w_i^q} \tag{7}$$

where $\gamma_q$ is the weight for visual feature $w^q$.

The next term $P(z_i = t|z_{L(i)} = r, z_{T(i)} = s)$ is the probability of state of $t$, given the states of the nodes on the left and above, i.e. $L(i)$ and $T(i)$. Let $n_{rst}$ be the number of observations with state $t$ whose the left and upper neighbor nodes' states are $r$ for $L(i)$ and $s$ for $T(i)$. The probability of generating state $t$ is given by:

$$P(z_i = t|z_{L(i)} = r, z_{T(i)} = s) = \frac{n_{rst} + \alpha' \beta_t}{\sum_{t'} n_{rst'} + \alpha'} \tag{8}$$

where $\beta_t$ refers to the weight of state $t$. This calculation follows the properties of Dirichlet distribution [5].

The last two terms $P(z_{R(i)}|z_i = t, z_{T(R(i))})$ and $P(z_{D(i)}|z_{L(D(i))}, z_i = t)$ are the probability of the states of the right and lower neighbor nodes $(R(i), D(i))$ given $z_i$. These two terms can be computed in a similar form as equation (8).

**Sampling $\beta$.** In the second stage, given the assignments $z = \{z_i\}$, we sample $\beta$ using the Dirichlet distribution as described in [5].

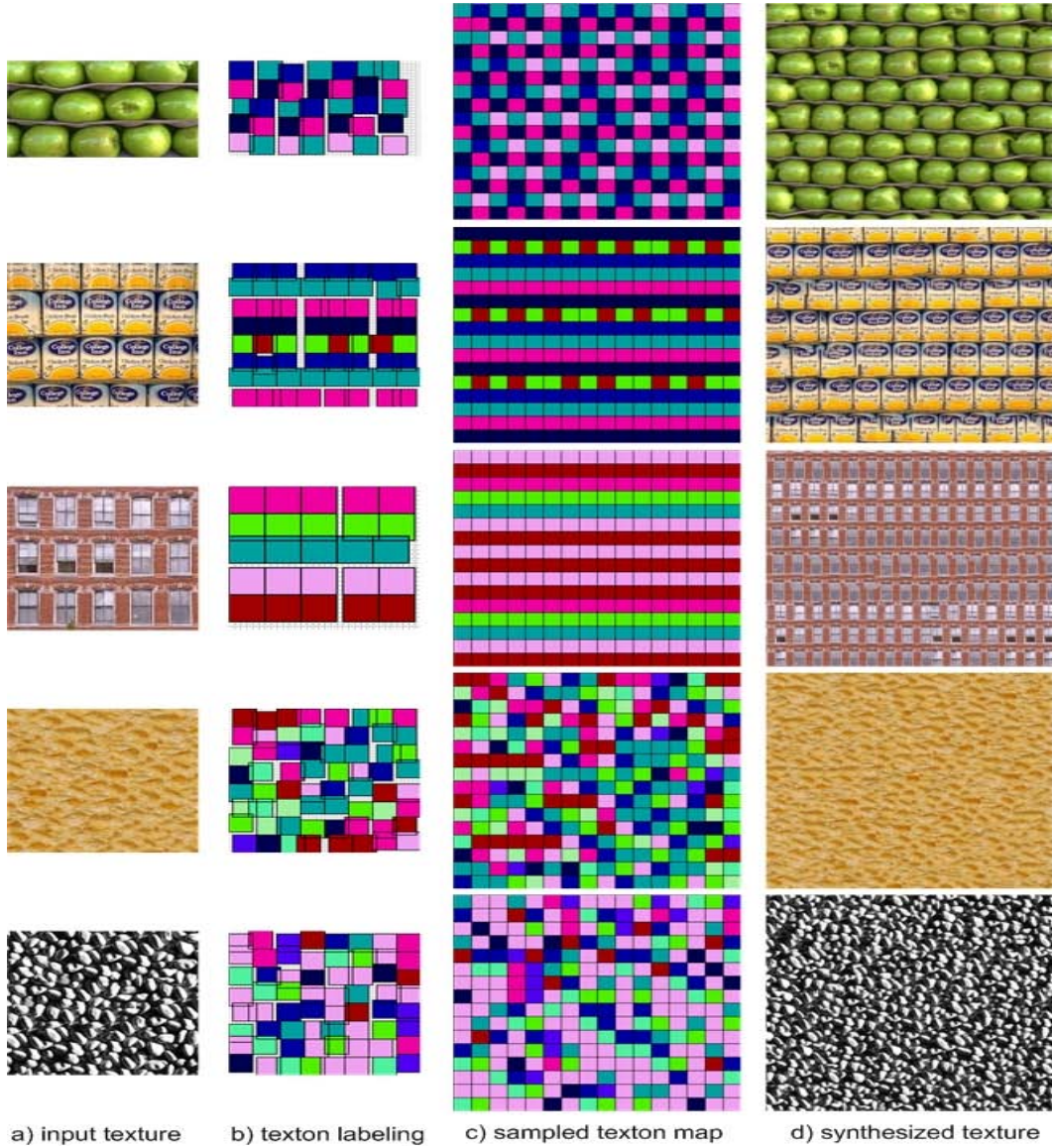

a) input texture    b) texton labeling    c) sampled texton map    d) synthesized texture

Figure 4: The color of rectangles in columns 2 and 3 correspond to the index (labeling) of textons which are represented by 24*24 image patches. The synthesized images are all 384*384 (16*16 textons /patches). Our method captures both stochastic textures (the last two rows) and more structured textures (the first three rows, see the horizontal and grided layouts). The inferred texton maps for structured textures are simpler (less states/textons) and more regular (less cluttered texton maps) than stochastic textures.

## 5 Texture Synthesis

Once the texton vocabulary and the transition matrix are learnt, the synthesis process first samples the latent texton labeling map according to the probability encoded in the transition matrix. But the HDP-2DHMM is generative only for image features, but not image intensity. To make it practical for image synthesis, image quilting [3] is integrated with the HDP-2DHMM. The final image is then generated by selecting image patches according to the texton labeling map. Image quilting is applied to select and stitch together all the patches in a top-left-to-bottom-right order so that the boundary inconsistency is minimized . The width of the overlap edge is 8 pixels. By contrast to [3] which need to search over all image patches to ensure high rendering quality, our method is only required to search the candidate patches within a local cluster. The HDP-2DHMM is capable of producing high rendering quality because the patches have been grouped based on visual features. Therefore, the synthesis cost is dramatically reduced. We show that the HDP-2DHMM is able to synthesize a

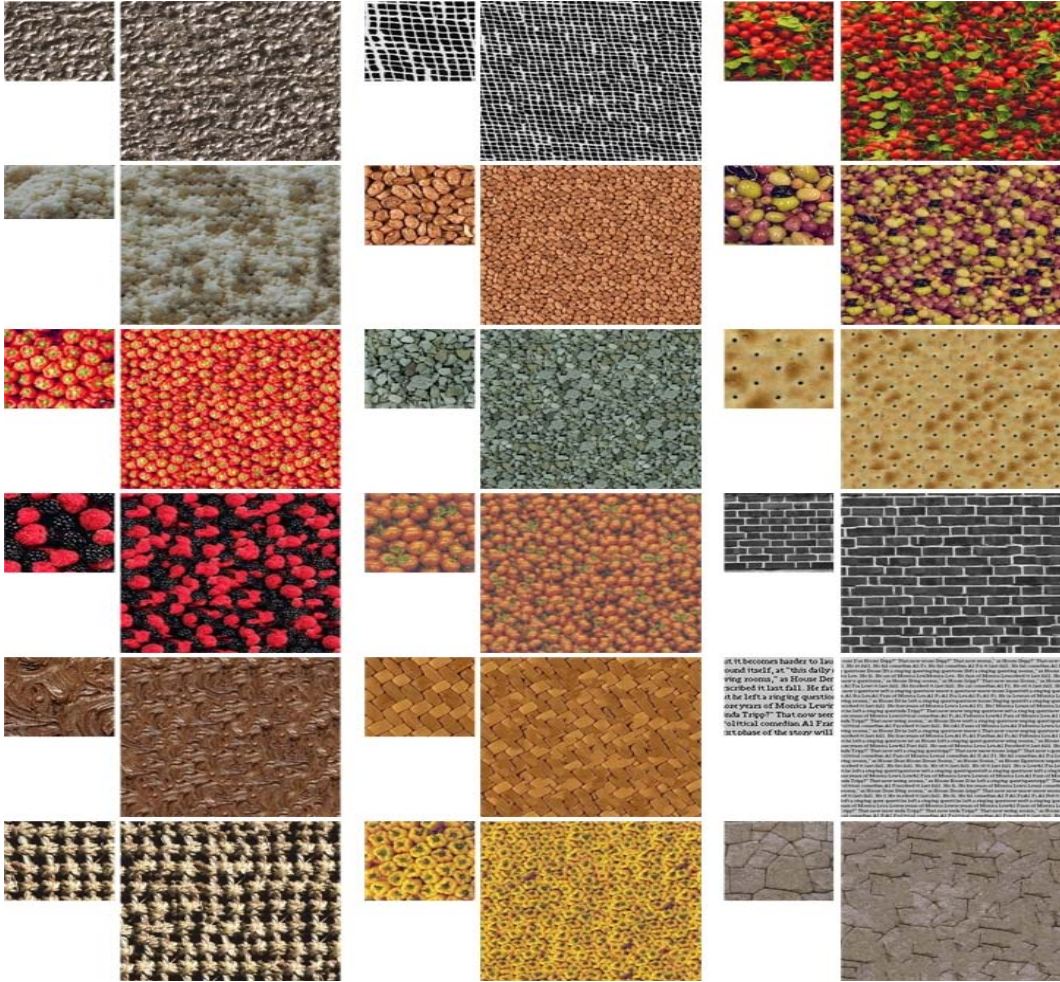

Figure 5: More synthesized texture images (for each pair, left is input texture, right is synthesized).

texture image with size of 384*384 and with comparable quality in one second (25 times faster than image quilting).

## 6 Experimental Results

### 6.1 Texture Learning and Synthesis

We use the texture images in [3]. The hyperparameters $\{\alpha, \alpha', \gamma\}$ are set to 10, 1, and 0.5, respectively. The image patch size is fixed to 24*24. All the parameter settings are identical for all images. The learning runs with 10 random initializations each of which takes about 30 sampling iterations to converge. A computer with 2.4 GHz CPU was used. For each image, it takes 100 seconds for learning and 1 second for synthesis (almost 25 times faster than [3]).

Figure (4) shows the inferred texton labeling maps, the sampled texton maps and the synthesized texture images. More synthesized images are shown in figure (5). The rendering quality is visually comparable with [3] (not shown) for both structured textures and stochastic textures. It is interesting to see that the HMM-HDP captures different types of texture patterns, such as vertical, horizontal and grided layouts. It suggests that our method is able to discover the semantic texture meaning by learning texton vocabulary and their spatial relations.

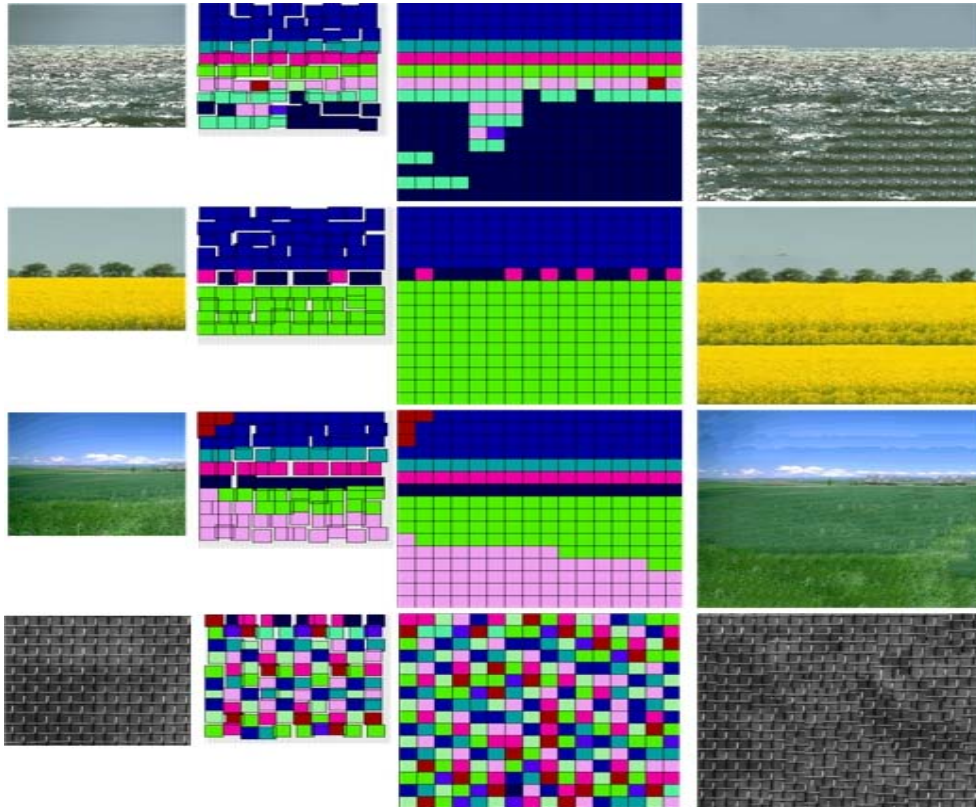

**Figure 6:** Image segmentation and synthesis. The first three rows show the HDP-2DHMM is able to segment images with mixture of textures and synthesize new textures. The last row shows a failure example where the texton is not well aligned.

## 6.2 Image Segmentation and Synthesis

We also apply the HDP-2DHMM to perform image segmentation and synthesis. Figure (6) shows several examples of natural images which contain mixture of textured regions. The segmentation results are represented by the inferred state assignments (the texton map). In figure (6), one can see that our method successfully divides images into meaningful regions and the synthesized images look visually similar to the input images. These results suggest that the HDP-2DHMM framework is generally useful for low-level vision problems. The last row in figure (6) shows a failure example where the texton is not well aligned.

## 7 Conclusion

This paper describes a novel nonparametric Bayesian method for textrure learning and synthesis. The 2D Hidden Markov Model (HMM) is coupled with the hierarchical Dirichlet process (HDP) which allows the number of textons and the complexity of transition matrix grows as the input texture becomes irregular. The HDP makes use of Dirichlet process prior which favors regular textures by penalizing the model complexity. This framework (HDP-2DHMM) learns the texton vocabulary and their spatial layout jointly and automatically. We demonstrated that the resulting compact representation obtained by the HDP-2DHMM allows fast texture synthesis (under one second) with comparable rendering quality to the state-of-the-art image-based rendering methods. Our results on image segmentation and synthesis suggest that the HDP-2DHMM is generally useful for further applications in low-level vision problems.

**Acknowledgments.** This work was supported by NGA NEGI-1582-04-0004, MURI Grant N00014-06-1-0734, ARDA VACE, and gifts from Microsoft Research and Google. Thanks to the anonymous reviewers for helpful feedback.

# References

[1] Y. N. Wu, S. C. Zhu, and C.-e. Guo, "Statistical modeling of texture sketch," in *ECCV '02: Proceedings of the 7th European Conference on Computer Vision-Part III*, 2002, pp. 240–254.

[2] S.-C. Zhu, C.-E. Guo, Y. Wang, and Z. Xu, "What are textons?" *International Journal of Computer Vision*, vol. 62, no. 1-2, pp. 121–143, 2005.

[3] A. A. Efros and W. T. Freeman, "Image quilting for texture synthesis and transfer," in *Siggraph*, 2001.

[4] A. Efros and T. Leung, "Texture synthesis by non-parametric sampling," in *International Conference on Computer Vision*, 1999, pp. 1033–1038.

[5] Y. W. Teh, M. I. Jordan, M. J. Beal, and D. M. Blei, "Hierarchical dirichlet processes," *Journal of the American Statistical Association*, 2006.

[6] M. J. Beal, Z. Ghahramani, and C. E. Rasmussen, "The infinite hidden markov model," in *NIPS*, 2002.

[7] S. C. Zhu, Y. Wu, and D. Mumford, "Filters, random fields and maximum entropy (frame): Towards a unified theory for texture modeling," *International Journal of Computer Vision*, vol. 27, pp. 1–20, 1998.

[8] A. Criminisi, P. Perez, and K. Toyama, "Region filling and object removal by exemplar-based inpainting," *IEEE Trans. on Image Processing*, 2004.

[9] J. Malik, S. Belongie, J. Shi, and T. Leung, "Textons, contours and regions: Cue integration in image segmentation," *IEEE International Conference on Computer Vision*, vol. 2, 1999.

[10] T. Leung and J. Malik, "Representing and recognizing the visual appearance of materials using three-dimensional textons," *International Journal of Computer Vision*, vol. 43, pp. 29–44, 2001.

[11] M. Varma and A. Zisserman, "Texture classification: Are filter banks necessary?" *IEEE Computer Society Conference on Computer Vision and Pattern Recognition*, vol. 2, 2003.

[12] Y. Liu, W.-C. Lin, and J. H. Hays, "Near regular texture analysis and manipulation," *ACM Transactions on Graphics (SIGGRAPH 2004)*, vol. 23, no. 1, pp. 368 – 376, August 2004.

[13] J. Hays, M. Leordeanu, A. A. Efros, and Y. Liu, "Discovering texture regularity as a higher-order correspondence problem," in *9th European Conference on Computer Vision*, May 2006.

[14] N. Jojic, B. J. Frey, and A. Kannan, "Epitomic analysis of appearance and shape," in *In ICCV*, 2003, pp. 34–41.

[15] A. Kannan, J. Winn, and C. Rother, "Clustering appearance and shape by learning jigsaws," in *In Advances in Neural Information Processing Systems*. MIT Press, 2007.

[16] J. J. Kivinen, E. B. Sudderth, and M. I. Jordan, "Learning multiscale representations of natural scenes using dirichlet processes," *IEEE International Conference on Computer Vision*, vol. 0, 2007.

[17] J. Domke, A. Karapurkar, and Y. Aloimonos, "Who killed the directed model?" in *IEEE Computer Society Conference on Computer Vision and Pattern Recognition*, 2008.

[18] L. Cao and L. Fei-Fei, "Spatially coherent latent topic model for concurrent object segmentation and classification," in *Proceedings of IEEE International Conference on Computer Vision*, 2007.

[19] X. Wang and E. Grimson, "Spatial latent dirichlet allocation," in *NIPS*, 2007.

[20] L.-Y. Wei and M. Levoy, "Fast texture synthesis using tree-structured vector quantization," in *SIGGRAPH '00: Proceedings of the 27th annual conference on Computer graphics and interactive techniques*, 2000, pp. 479–488.

[21] J. Winn, A. Criminisi, and T. Minka, "Object categorization by learned universal visual dictionary," in *Proceedings of the Tenth IEEE International Conference on Computer Vision*, 2005.

